# Efficient Sampling for Bipartite Matching Problems

**Maksims N. Volkovs**
University of Toronto
mvolkovs@cs.toronto.edu

**Richard S. Zemel**
University of Toronto
zemel@cs.toronto.edu

## Abstract

Bipartite matching problems characterize many situations, ranging from ranking in information retrieval to correspondence in vision. Exact inference in real-world applications of these problems is intractable, making efficient approximation methods essential for learning and inference. In this paper we propose a novel *sequential matching* sampler based on a generalization of the Plackett-Luce model, which can effectively make large moves in the space of matchings. This allows the sampler to match the difficult target distributions common in these problems: highly multimodal distributions with well separated modes. We present experimental results with bipartite matching problems—ranking and image correspondence—which show that the sequential matching sampler efficiently approximates the target distribution, significantly outperforming other sampling approaches.

## 1 Introduction

Bipartite matching problems (BMPs), which involve mapping one set of items to another, are ubiquitous, with applications ranging from computational biology to information retrieval to computer vision. Many problems in these domains can be expressed as a bipartite graph, with one node for each of the items, and edges representing the compatibility between pairs.

In a typical BMP a set of labeled instances with target matches is provided together with feature descriptions of the items. The features for any two items do not provide a natural measure of compatibility between the items, i.e., should they be matched or not. Consequently the goal of learning is to create a mapping from the item features to the target matches such that when an unlabeled instance is presented the same mapping can be applied to accurately infer the matches. Probabilistic formulations of this problem, which involve specifying a distribution over possible matches, have become increasingly popular, e.g., [23, 26, 1], and these models have been applied to problems ranging from preference aggregation in social choice and information retrieval [7, 13] to multiple sequence protein alignment in computational biology [24, 27].

However, exact learning and inference in real-world applications of these problems quickly become intractable because the state space is typically factorial in the number of items. Approximate inference methods are also problematic in this domain. Variational approaches, in which aspects of the joint distribution are treated independently, may miss important contingencies in the joint. On the other hand sampling is hard, plagued by the multimodality and strict constraints inherent in discrete combinatorial spaces.

Recently there has been a flurry of new methods for sampling for bipartite matching problems. Some of these have strong theoretical properties [10, 9], while others are appealingly simple [6, 13]. However, to the best of our knowledge, even for simple versions of bipartite matching problems, no efficient sampler exists. In this paper we propose a novel Markov Chain Monte Carlo (MCMC) method applicable to a wide subclass of BMPs. We compare the efficiency and performance of our sampler to others on two applications.

## 2   Problem Formulation

A standard BMP consists of the two sets of $N$ items $U = \{u_1, ..., u_N\}$ and $V = \{v_1, ..., v_N\}$. The goal is to find an assignment of the items so that every item in $U$ is matched to exactly one item in $V$ and no two items share the same match. In this problem an assignment corresponds to a permutation $\pi$ where $\pi$ is a bijection $\{1, ..., N\} \rightarrow \{1, ..., N\}$, mapping each item in $U$ to its match in $V$; we use the terms assignment and permutation interchangeably. We define $\pi(i) = j$ to denote the index of a match $v_{\pi(i)} = v_j$ for item $u_i$ in $\pi$ and use $\pi^{-1}(j) = i$ to denote the reverse. Permutations have a useful property that any subset of the permutation also constitutes a valid permutation with respect to the items in the subset. We will utilize this property in later sections; here we introduce the notation. Given a full permutation $\pi$ we define $\pi_{1:t}$ ($\pi_{1:0} = \emptyset$) as a partial permutation of only the first $t$ items in $U$.

To express uncertainty over assignments, we use the standard Gibbs form to define the probability of a permutation $\pi$:

$$P(\pi|\theta) = \frac{1}{Z(\theta)} \exp(-E(\pi, \theta)) \qquad Z(\theta) = \sum_\pi \exp(-E(\pi, \theta)) \tag{1}$$

where $\theta$ is the set of model parameters and $E(\pi, \theta)$ is the energy. We assume, without loss of generality, that the energy $E(\pi, \theta)$ is given by a sum of single and/or higher order potentials.

Many important problems can be formulated in this form. For example, in information retrieval the crucial problem of learning a ranking function can be modeled as a BMP [12, 26]. In this domain $U$ corresponds to a set of documents and $V$ to a set of ranks. The energy of a given assignment is typically formulated as a combination of ranks and the model's output from the query-document features. For example in [12] the energy is defined as:

$$E(\pi, \theta) = -\frac{1}{N} \sum_{i=1}^{N} \theta_i (N - \pi(i) + 1) \tag{2}$$

where $\theta_i$ is a score assigned by the model to $u_i$. Similarly, in computer vision the problem of finding a correspondence between sets of images can be expressed as a BMP [5, 3, 17]. Here $U$ and $V$ are typically sets of points in the two images and the energy is defined on the feature descriptors of these points. For example in [17] the energy is given by:

$$E(\pi, \theta) = \frac{1}{|\psi|} \sum_{i=1}^{N} \left\langle \theta, (\psi_i^u - \psi_{\pi(i)}^v)^2 \right\rangle \tag{3}$$

where $\psi_i^u$ and $\psi_{\pi(i)}^v$ are feature descriptors for points $u_i$ and $v_{\pi(i)}$. Finally, some clustering problems can also be expressed in the form of a BMP [8]. It is important to note here that for all models where the energy is additive we can compute the energy $E(\pi_{1:t}, \theta)$ for any partial permutation $\pi_{1:t}$ by summing the potentials only over the $t$ assignments in $\pi_{1:t}$. For instance for the energy in Equation 3, $E(\pi_{1:t}, \theta) = \frac{1}{|\psi|} \sum_{i=1}^{t} \left\langle \theta, (\psi_i^u - \psi_{\pi_{1:t}(i)}^v)^2 \right\rangle$ with $E(\pi_{1:0}, \theta) = 0$.

Learning in these models typically involves maximizing the log probability of the correct match as a function of $\theta$. To do this one generally needs to find the gradient of the log probability with respect to $\theta$: $\frac{\partial \log(P(\pi|\theta))}{\partial \theta} = -\frac{\partial E(\pi, \theta)}{\partial \theta} - \frac{\partial \log(Z(\theta))}{\partial \theta}$. Unfortunately, computing the gradient with respect to the partition function requires a summation over $N!$ valid assignments, which very quickly becomes intractable. For example for $N = 20$ finding $\frac{\partial \log(Z(\theta))}{\partial \theta}$ requires over $10^{17}$ summations. Thus effective approximation techniques are necessary to learn such models.

A particular instance of BMP that has been studied extensively is the *maximum weight bipartite matching problem (WBMP)*. In WBMP the energy is reduced to only the single potential $\phi$:

$$E^{unary}(\pi, \theta) = \sum_i \phi(u_i, v_{\pi(i)}, \theta) \tag{4}$$

Equations 2 and 3 are both examples of WBMP energies. Finding the assignment with the maximum energy is tractable and can be solved in $O(N^3)$ [16]. Determining the partition function in a WBMP is equivalent to finding the permanent of the edge weight matrix (defined by the unary potential), a well-known $\#P$ problem [25]. The majority of the proposed samplers are designed for

WBMPs and cannot be applied to the more general BMPs where the energy includes higher order potentials. However, distributions based on higher order potentials allow greater flexibility and have been actively used in problems ranging from computer vision and robotics [20, 2] to information retrieval [19, 26]. There is thus an evident need to develop an effective sampler applicable to any BMP distribution.

# 3 Related Approaches

In this section we briefly describe existing sampling approaches, some of which have been developed specifically for bipartite matching problems while others come from matrix permanent research.

## 3.1 Gibbs Sampling

Gibbs and block-Gibbs sampling can be applied straightforwardly to sample from distributions defined by Equation 1. To do that we start with some initial assignment $\pi$ and consider a subset of items in $U$; for illustration purposes we will use two items $u_i$ and $u_j$. Given the selected subset of items the Gibbs sampler considers all possible assignment swaps within this subset. In our example there are only two possibilities: leave $\pi$ unchanged or swap $\pi(i)$ with $\pi(j)$ to produce a new permutation $\pi'$. Conditioned on the assignment of all the other items in $U$ that were not selected, the probability of each permutation is:

$$p(\pi'|\pi_{\setminus\{i,j\}}) = \frac{\exp(-E(\pi',\theta))}{\exp(-E(\pi,\theta)) + \exp(-E(\pi',\theta))} \qquad p(\pi|\pi_{\setminus\{i,j\}}) = 1 - p(\pi'|\pi_{\setminus\{i,j\}})$$

where $\pi_{\setminus\{i,j\}}$ is permutation $\pi$ with $u_i$ and $u_j$ removed. We sample using these probabilities to either stay at $\pi$ or move to $\pi'$, and repeat the process.

Gibbs sampling has been applied to a wide range of energy-based probabilistic models. It is often found to mix very slowly and to get trapped in local modes [22]. The main reason for this is that the path from one probable assignment to another using only pairwise swaps is likely to go through regions that have very low probability [5]. This makes it very unlikely that those moves will be accepted, which typically traps the sampler in one mode. Thus, the local structure of the Gibbs sampler is likely to be inadequate for problems of the type considered here, in which several probable assignments will produce well-separated modes.

## 3.2 Chain-Based Approaches

Chain-based methods extend the assignment swap idea behind the Gibbs sampler to generate samples more efficiently from WBMP distributions. Instead of randomly choosing subsets of items to swap, chain-based method generate a sequence (chain) of interdependent swaps. Given a (random) starting permutation $\pi$, an item $u_i$ (currently matched with $v_{\pi(i)}$) is selected at random and a new match $v_j$ is proposed with probability $p(u_i, v_j|\theta)$ where $p$ depends on the unary potential $\phi(u_i, v_j, \theta)$ in the WBMP energy (see Equation 4). Now, assuming that the match $\{u_i, v_j\}$, is selected, matches $\{u_i, v_{\pi(i)}\}$ and $\{u_{\pi^{-1}(j)}, v_j\}$ are removed from $\pi$ and $\{u_i, v_j\}$ is added to make $\pi'$. After this change $u_{\pi^{-1}(j)}$ and $v_{\pi(i)}$ are no longer matched to any item so $\pi'$ is a partial assignment. The procedure then finds a new match for $u_{\pi^{-1}(j)}$ using $p$. This chain-like match sampling is repeated either until $\pi'$ is a complete assignment or a termination criteria is reached.

Several chain-based methods have been proposed including the chain flipping approach [5] and the Markov Chain approach [11]. Dellaert et al., [5] empirically demonstrated that the chain flipping sampler can mix better than the Gibbs sampler when applied to multimodal distributions. However, chain-based methods also have several drawbacks that significantly affect their performance. First, unlike the Gibbs sampler which always maintains a valid assignment, the intermediate assignments $\pi'$ in chain-based methods are incomplete. This means that the chain either has to be run until a valid assignment is generated [5] or terminated early and produce an incomplete assignment [11]. In the first case the sampler has a non-deterministic run-time whereas in the second case the incomplete assignment can not be taken as a valid sample from the model. Finally, to the best of our knowledge no chain-based method can be applied to general BMPs because they are specifically designed for $E^{unary}$ (see Equation 4).

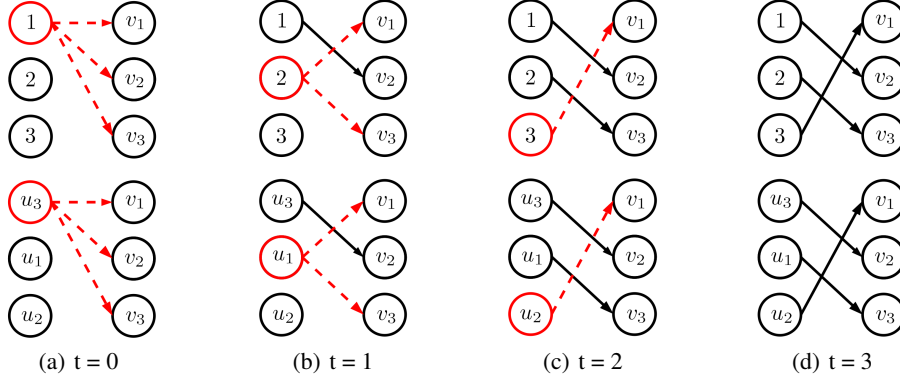

| (a) t = 0 | (b) t = 1 | (c) t = 2 | (d) t = 3 |

Figure 1: Top row: Plackett-Luce generative process viewed as rank matching. Bottom row: sequential matching procedure. Items are $U = \{u_1, u_2, u_3\}$ and $V = \{v_1, v_2, v_3\}$; the reference permutation is $\sigma = \{2, 3, 1\}$. The proposed matches are shown in red dotted arrows and accepted matches in black arrows.

### 3.3 Recursive Partitioning Algorithm

The recursive partitioning [10] algorithm was developed to obtain exact samples from the distribution for WBMP. This method is considered to be the state-of-the-art in matrix permanent research and to the beset of our knowledge has the lowest expected run time. Recursive partitioning proceeds by splitting the space of all valid assignments $\Omega$ into $K$ subsets $\Omega_1, ..., \Omega_K$ with corresponding partition functions $Z_1, ..., Z_K$. It then samples one of these subsets and repeats the partitioning procedure recursively, generating exact samples from a WBMP distribution.

Despite strong theoretical guarantees the recursive partitioning procedure has a number of limitations that significantly affect its applicability. First, the running time of the sampler is non-deterministic as the algorithm has to be restarted every time a sample falls outside of $\Omega$. The probability of restart increases with $N$ which is an undesirable property especially for training large models where one typically needs to have precise control over the time spent in each training phase. Moreover, this algorithm is also specific to WBMP and cannot be generalized to sample from arbitrary BMP distributions with higher order potentials.

### 3.4 Plackett-Luce Model

Our proposed sampling approach is based on a generalization of the well-established Plackett-Luce model [18, 14], which is a generative model for permutations. Given a set of items $V = \{v_1, ..., v_N\}$, a Plackett-Luce model is parametrized by a set of weights (one per item) $W = \{w_1, ..., w_N\}$. Under this model a permutation $\pi$ is generated by first selecting item $v_{\pi(1)}$ from the set of $N$ items and placing it in the first position, then selecting $v_{\pi(2)}$ from the remaining $N - 1$ items and placing it the second position, and so on until all $N$ items are placed. The probability of $\pi$ under this model is given by:

$$Q(\pi) = \frac{\exp(w_{\pi(1)})}{\sum_{i=1}^{N} \exp(w_{\pi(i)})} \times \frac{\exp(w_{\pi(2)})}{\sum_{i=2}^{N} \exp(w_{\pi(i)})} \times ... \times \frac{\exp(w_{\pi(N)})}{\exp(w_{\pi(N)})} \quad (5)$$

Here $\frac{\exp(w_{\pi(t)})}{\exp(\sum_{i=t}^{N} w_{\pi(i)})}$ is the probability of choosing the item $v_{\pi(t)}$ out of the $N - t + 1$ remaining items. It can be shown that $Q$ is a valid distribution with $\sum_{\pi} Q(\pi) = 1$. Moreover, note that it is very easy to draw samples from $Q$ by applying the sequential procedure described above. In the next section we show how this model can be generalized to draw samples from any BMP distribution.

## 4 Sampling by Sequentially Matching Vertices

In this section we introduce a class of proposal distributions that can be effectively used in conjunction with the Metropolis-Hastings algorithm to obtain samples from a BMP distribution. Our approach is based on the observation that the sequential procedure behind the Plackett-Luce model can also be extended to generate matches between item sets. Instead of placing items into ranked positions we can think of the Plackett-Luce generative process as sequentially matching ranks to the items in $V$, as illustrated in the top row of Figure 1. To generate the permutation $\pi = \{3, 1, 2\}$ the Plackett-Luce model first matches rank 1 with $v_{\pi(1)} = v_2$ then rank 2 with $v_{\pi(2)} = v_3$ and finally rank 3 with $v_{\pi(3)} = v_1$. Taking this one step further we can replace ranks with a general item set

$U$ and repeat the same process. Unlike ranks, items in $U$ do not have a natural order so we use a *reference permutation* $\sigma$, which specifies the order in which items in $U$ are matched. We refer to this procedure as *sequential matching*. The bottom row of Figure 1 illustrates this process.

Formally the sequential matching process proceeds as follows: given some reference permutation $\sigma$, we start with an empty assignment $\pi_{1:0} = \emptyset$. Then at each iteration $t = 1, ..., N$ the corresponding item $u_{\sigma(t)}$ gets matched with one of the $V \setminus \pi_{1:t-1}$ items, where $V \setminus \pi_{1:t-1} = \{v_{j_t}, ..., v_{j_N}\}$ denotes the set of items not matched in $\pi_{1:t-1}$. Note that similarly to the Plackett-Luce model, $|V \setminus \pi_{1:t-1}| = N - t + 1$ so at each iteration, $u_{\sigma(t)}$ will have $N - t + 1$ left over items in $V \setminus \pi_{1:t-1}$ to match with. We define the conditional probability of each such match to be $p(v_j|u_{\sigma(t)}, \pi_{1:t-1})$, $\sum_{v_j \in V \setminus \pi_{1:t-1}} p(v_j|u_{\sigma(t)}, \pi_{1:t-1}) = 1$. After $N$ iterations the permutation $\pi_{1:N} = \pi$ is produced with probability:

$$Q(\pi|\sigma) = \prod_{t=1}^{N} p(v_{\pi(\sigma(t))}|u_{\sigma(t)}, \pi_{1:t-1}) \tag{6}$$

where $v_{\pi(\sigma(t))}$ is a match for $u_{\sigma(t)}$ in $\pi$. The conditional match probabilities depend on both the current item $u_{\sigma(t)}$ and on the partial assignment $\pi_{1:t-1}$. Introducing this dependency generalizes the Plackett-Luce model which only takes into account that the items in $\pi_{1:t-1}$ are already matched but does not take into account *how* these items are matched. This dependency becomes very important when the energy contains pairwise and/or higher order potentials as it allows us to compute the change in energy for each new match, in turn allowing for close approximations to the target BMP distribution.

We can show that the distribution $Q$ defined by the $p$'s is a valid distribution over assignments:

**Proposition 1** *For any reference permutation $\sigma$ and any choice of matching probabilities that satisfy $\sum_{v_j \in V \setminus \pi_{1:t-1}} p(v_j|u_{\sigma(t)}, \pi_{1:t-1}) = 1$, the distribution given by: $Q(\pi|\sigma) = \prod_{t=1}^{N} p(v_{\pi(\sigma(t))}|u_{\sigma(t)}, \pi_{1:t-1})$ is a valid probability distribution over assignments.*[1]

The important consequence of this proposition is that it allows us to work with a very rich class of matching probabilities with arbitrary dependencies and still obtain a valid distribution over assignments with a simple way to generate exact samples from it. This opens many avenues for tailoring proposal distributions for MCMC applications to specific BMPs. In the next section we propose one such approach.

### 4.1 Proposal Distribution

Given the general matching probabilities the goal is to define them so that the resulting proposal distribution $Q$ matches the target distribution as closely as possible. One potential way of achieving this is through the partial energy $E(\pi_{1:t}, \theta)$ (see Section 2). The partial energy ignores all the items that are not matched in $\pi_{1:t}$ and thus provides an estimate of the "current" energy at each iteration $t$. Using partial energies we can also find the changes in energy when a given item is matched. Given that our goal is to explore low-energy (high-probability) modes we define the matching probabilities as:

$$p(v_j|u_{\sigma(t)}, \pi_{1:t-1}) = \frac{\exp(-E(H(v_j, u_{\sigma(t)}, \pi_{1:t-1}), \theta))}{Z_t(u_{\sigma(t)}, \pi_{1:t-1})}$$
$$Z_t(u_{\sigma(t)}, \pi_{1:t-1}) = \sum_{v_j \in V \setminus \pi_{1:t-1}} \exp(-E(H(v_j, u_{\sigma(t)}, \pi_{1:t-1}), \theta)) \tag{7}$$

where $H(v_j, u_{\sigma(t)}, \pi_{1:t-1})$ is the resulting partial assignment after match $\{u_{\sigma(t)}, v_j\}$ is added to $\pi_{1:t-1}$. The normalizing constant $Z_t$ ensures that the probabilities sum to 1, which is the necessary condition for Proposition 1 to apply. It is useful to rewrite the matching probabilities as:

$$p(v_j|u_{\sigma(t)}, \pi_{1:t-1}) = \frac{\exp(-E(H(v_j, u_{\sigma(t)}, \pi_{1:t-1}), \theta) + E(\pi_{1:t-1}, \theta))}{Z_t^*(u_{\sigma(t)}, \pi_{1:t-1})}$$
$$Z_t^*(u_{\sigma(t)}, \pi_{1:t-1}) = \sum_{v_j \in V \setminus \pi_{1:t-1}} \exp(-E(H(v_j, u_{\sigma(t)}, \pi_{1:t-1}), \theta) + E(\pi_{1:t-1}, \theta))$$

Adding $E(\pi_{1:t-1}, \theta)$ to each item's energy does not change the probabilities because this term cancels out during normalization (but it does change the partition function, denoted by $Z_t^*$ here). However, in this form we see that $p(v_j|u_{\sigma(t)}, \pi_{1:t-1})$ is directly related to the change in the partial energy

from $\pi_{1:t-1}$ to $H(v_j, u_{\sigma(t)}, \pi_{1:t-1})$ – the larger the change the bigger the resulting probability will be. Thus, the matching choices will be made solely based on the changes in the partial energy. Reorganizing the terms yields the proposal distribution:

$$Q(\pi|\sigma) = \frac{\exp(-E(\pi_{1:1}, \theta) + E(\pi_{1:0}, \theta))}{Z_1^*(u_{\sigma(1)}, \pi_{1:0})} \times ... \times \frac{\exp(-E(\pi_{1:N}, \theta) + E(\pi_{1:N-1}, \theta))}{Z_N^*(u_{\sigma(N)}, \pi_{1:N-1})} = \frac{\exp(-E(\pi, \theta))}{Z^*(\pi, \sigma)}$$

Here $Z^*(\pi, \sigma)$ is the normalization factor which depends both on the reference permutation $\sigma$ and the generated assignment $\pi$. The resulting proposal distribution is essentially a renormalized version of the target distribution. The numerator remains the exponent of the energy but the denominator is no longer a constant; rather it is a function which depends on the generated assignment and the reference permutation. Note that the proposal distribution defined above can be used to generate samples for *any* target distribution with arbitrary energy consisting of single and/or higher order potentials. To the best of our knowledge aside from the Gibbs sampler this is the only sampling procedure that can be applied to arbitrary BMP distributions.

## 4.2 Temperature and Chain Properties

Acceptance rate, a key property of any sampler, is typically controlled by a parameter which either shrinks or expands the proposal distribution. To achieve this effect with the sequential matching model we introduce an additional parameter $\rho$ which we refer to as *temperature*: $p(v_j|u_{\sigma(t)}, \pi_{1:t-1}, \rho) \propto \exp(-E(H(v_j, u_{\sigma(t)}, \pi_{1:t-1}), \theta)/\rho)$. Decreasing $\rho$ leads to sharp proposal distributions typically highly skewed towards one specific assignment, while increasing $\rho$ makes the proposal distribution approach the uniform distribution. By adjusting $\rho$ we can control the range of the proposed moves therefore controlling the acceptance rate.

To ensure that the SM sampler converges to the required distribution we demonstrate that it satisfies the three requisite properties: detailed balance, ergodicity, and aperiodicity [15]. The detailed balance condition is satisfied because every Metropolis-Hastings algorithm satisfies detailed balance. Ergodicity follows from the fact that the insertion probabilities are always strictly greater than 0. Therefore any $\pi$ is reachable from any $\sigma$ in one proposal cycle. Finally, aperiodicity follows from the fact that the chain allows self-transitions.

## 4.3 Reference Permutation

Fixing the reference permutation $\sigma$ yields a state independent sampler. Empirically we found that setting $\sigma$ to the MAP permutation gives good performance for WBMP problems. However, for the general energy based distributions considered here finding the MAP state can be very expensive and in many cases intractable. Moreover, even if MAP can be found efficiently there is still no guarantee that using it as the reference permutation will lead to a good sampler. To avoid these problems we use a state dependent sampler where the reference permutation $\sigma$ is updated every time a sample gets accepted. In the matching example (bottom row of Figure 1) if the new match at $t = 3$ is accepted then $\sigma$ would be updated to $\{3, 1, 2\}$. Empirically we found the state dependent sampler to be more stable, with consistent performance across different random initializations of the reference permutation. Algorithm 1 summarizes the Metropolis-Hastings procedure for the state dependent sequential matching sampler.

---
**Algorithm 1** Sequential Matching (SM)
---
**Input:** $\sigma$, $M$, $\rho$
**for** $m = 1$ **to** $M$ **do**
    Initialize $\pi_{1:0} = \emptyset$
    **for** $t = 1$ **to** $N$ **do** {generate sample from $Q(\cdot|\sigma)$}
        Find a match $v_j$ for $u_{\sigma(t)}$ using:
            $p(v_j|u_{\sigma(t)}, \pi_{1:t-1}, \rho)$
        Add $\{u_{\sigma(t)}, v_j\}$ to $\pi_{1:t-1}$ to get $\pi_{1:t}$
    **end for**
    Calculate forward probability:
        $Q(\pi|\sigma) = \prod_{t=1}^{N} p(v_{\pi(\sigma(t))}|u_{\sigma(t)}, \pi_{1:t-1}, \rho)$
    Calculate backward probability:
        $Q(\sigma|\pi) = \prod_{t=1}^{N} p(v_{\sigma(\pi(t))}|u_{\pi(t)}, \sigma_{1:t-1}, \rho)$
    **if** $Uniform(0,1) < \frac{\exp(-E(\pi,\theta))Q(\sigma|\pi)}{\exp(-E(\sigma,\theta))Q(\pi|\sigma)}$ **then**
        $\sigma \leftarrow \pi$
    **end if**
**end for**
**Return:** $\sigma$

---

## 5 Experiments

To test the sequential matching sampling approach we conducted extensive experiments. We considered document ranking and image matching, two popular applications of BMP; and for the sake of

Table 1: Average Hellinger distances for learning to rank (left half) and image matching (right half) problems. Statistically significant results are underlined. Note that Hellinger distances for $N = 8$ are not directly comparable to those for $N = 25, 50$ since approximate normalization is used for $N > 8$. For $N = 50$ we were unable to get a single sample from the RP sampler for any $c$ in the allocated time limit (over 5 minutes).

| | Learning to Rank | | | | | Image Matching | | | | |
|---|---|---|---|---|---|---|---|---|---|---|
| | c = 20 | c = 40 | c = 60 | c = 80 | c = 100 | c = 0.2 | c = 0.4 | c = 0.6 | c = 0.8 | c = 1 |
| **N = 8:** | | | | | | | | | | |
| GB | 0.7948 | 0.6211 | **0.4635** | **0.4218** | 0.3737 | **0.9108** | 0.8868 | 0.8320 | **0.7616** | **0.6533** |
| CF | 0.9012 | 0.8987 | 0.8887 | 0.8714 | 0.8748 | 0.9112 | 0.8882 | 0.8336 | 0.7672 | 0.6623 |
| RP | 0.7945 | 0.6209 | 0.4629 | 0.4986 | 0.3734 | 0.9110 | 0.8870 | 0.8312 | 0.7623 | 0.6548 |
| SM | **0.7902** | **0.6188** | 0.4636 | 0.4474 | **0.3725** | 0.9109 | **0.8866** | **0.8307** | 0.7621 | 0.6557 |
| **N = 25:** | | | | | | | | | | |
| GB | 0.9533 | 0.9728 | 0.9646 | 0.9449 | 0.9486 | 0.7246 | 0.8669 | 0.9902 | 0.9960 | 0.9976 |
| CF | 0.9767 | 0.9990 | 0.9937 | 0.9953 | 0.9781 | 0.7243 | 0.8675 | 0.9904 | 0.9950 | 0.9807 |
| RP | 0.9533 | 0.9728 | 0.9694 | 0.9462 | 0.9673 | 0.7279 | 0.9788 | 0.9896 | 0.9988 | 0.9969 |
| SM | **0.1970** | **0.1937** | **0.2899** | **0.4166** | **0.3858** | 0.7234 | **0.8471** | **0.8472** | **0.6350** | **0.5576** |
| **N = 50:** | | | | | | | | | | |
| GB | 0.9983 | 0.9991 | 0.9988 | 0.9974 | 0.9985 | 0.6949 | 0.9646 | 1.0000 | 1.0000 | 1.0000 |
| CF | 0.9841 | 0.9995 | 0.9993 | 0.9906 | 0.9305 | 0.6960 | 0.9635 | 1.0000 | 1.0000 | 0.9992 |
| SM | **0.1617** | **0.2335** | **0.3462** | **0.4931** | **0.4895** | 0.6941 | **0.9243** | **0.7016** | **0.3550** | **0.1677** |

comparison we concentrated on WBMP, as most of the methods cannot be applied to general BMP problems. When comparing the samplers we concentrated on evaluating how well the Monte Carlo estimates of probabilities produced by the samplers approximate the true distribution $P$. When target probabilities are known this method of evaluation provides a good estimate of performance since the ultimate goal of any sampler is to approximate $P$ as closely as possible.

For all experiments the Hellinger distance was used to compare the true distributions with the approximations produced by samplers. We chose this metric because it is symmetric and bounded. Furthermore it avoids the $\log(0)$ problems that arise in cross entropy measures. For any two distributions $P$ and $Q$ the Hellinger distance is given by $D = (1 - (\sum_\pi P(\pi)Q(\pi))^{1/2})^{1/2}$. Note that $0 \le D \le 1$ where 0 indicates that $P = Q$. Computing $D$ exactly quickly becomes intractable as the number of items grows. To overcome this problem we note that if a given permutation $\pi$ is not generated by any of the samplers then the term $P(\pi)Q(\pi)$ is 0 and does not affect the resulting estimate of $D$ for any sampler. Hence we can locally approximate $D$ up to a constant for all samplers by changing Equation 1 to: $P(\pi|\theta) \approx \frac{\exp(-E(\pi,\theta))}{\sum_{\pi' \in \Omega^*} \exp(-E(\pi',\theta))}$, where $\Omega^*$ is the union of all distinct permutations produced by the samplers. The Hellinger distance is then estimated with respect to $\Omega^*$. For all experiments we ran the samplers on small ($N = 8$), medium ($N = 25$) and large ($N = 50$) scale problems. The sampling chains for each method were run in parallel using 4 cores; the use of multiprocessor boards such as GPUs allows our method to scale to large problems. We compare the SM approach with Gibbs (GB), chain flipping (CF) and recursive partitioning (RP) samplers. To run RP we used the code available from the author's webpage. These methods cover all of the primary leading approaches in WBMP and matrix permanent research.

Since any valid sampler will eventually produce samples from the target distribution, we tested the methods with short chain lengths. This regime also simulates real applications of the methods where, due to computational time limits, the user is typically unable to run long chains. Note that this is especially relevant if the distributions are being sampled as an inner loop during parameter optimization. Furthermore, to make comparisons fair we used the block GB sampler with the block size of 7 (the largest computationally feasible size) as the reference point. We used $2N$ swaps for each GB chain, setting the number of iterations for other methods to match the total run time for GB (for all experiments the difference in running times between GB and SM did not exceed $10\%$). The run times of the CF and RP methods are difficult to control as they are non-deterministic. To deal with this we set an upper-bound on the running time (consistent with the other methods) after which CF and RP were terminated. Finally, the temperature for SM was chosen in the $[0.1, 1]$ interval to keep the acceptance rate approximately between $20\%$ and $60\%$.

## 5.1 Learning to Rank

For a learning to rank problem we used the Yahoo! Learning To Rank dataset [4]. For each query the distribution over assignments was parametrized by the energy given in Equation 2. Here $\theta_i$

is the output of the neural network scoring function trained on query-document features. After pretraining the network on the full dataset we randomly selected 50 queries with $N = 8, 25, 50$ documents and used GB, CF, RP and SM methods to generate 1000 samples for each query. To gain insight into sampling accuracy we experimented with different distribution shapes by introducing an additional scaling constant $c$ so that $P(\pi|\theta, c) \propto \exp(-c \times E(\pi, \theta))$. In this form $c$ controls the "peakiness" of the distribution with large values resulting in highly peaked distributions; we used $c \in \{20, 40, 60, 80, 100\}$.

The left half of Table 1 shows Hellinger distances for $N = 8, 25, 50$, averaged across the 50 queries.[2] From the table it is seen that all the samplers perform equally well when the number of items is small ($N = 8$). However, as the number of items increases SM significantly outperforms all other samplers. Throughout experiments we found that the CF and RP samplers often reached the allocated time limit and had to be forced to terminate early. For $N = 50$ we were unable to get a single sample from the RP sampler after running it for over 5 minutes. This is likely due to the fact that at each matching step $t = 1, ..., N$ the RP sampler has a non-zero probability of failing (rejecting). Consequently the total rejection probability increases linearly with the number of items $N$. Even for $N = 25$ we found the RP sampler to reject over 95% of the time. This further suggests that approaches with non-deterministic run times are not suitable for this problem because their worst-case performance can be extremely slow. Overall, the results indicate that SM can produce higher quality samples more rapidly, a crucial property for learning large-scale models.

## 5.2 Image Matching

For an image matching task we followed the framework of Petterson et al. [17]. Here, we used the Giraffe dataset [21] which is a video sequence of a walking giraffe. From this data we randomly selected 50 pairs of images that were at least 20 frames apart. Using the available set of 61 hand labeled points we then randomly selected three sets of correspondence points for each image pair, containing 8, 25 and 50 points respectively, and extracted SIFT feature descriptors at each point. The target distribution over matchings was parametrized by the energy given by Equation 3 where $\psi$'s are the SIFT feature descriptors. We also experimented with different scale settings: $c \in \{0.2, 0.4, 0.6, 0.8, 1\}$. Figure 2 shows an example pair of images with 25 labeled points and the inferred MAP assignment.

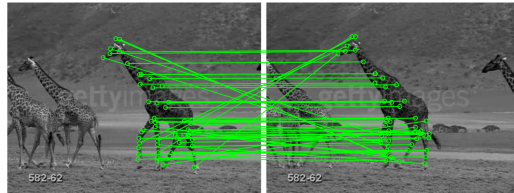

The results for $N = 8, 25, 50$ are shown in in the right half of Table 1. We see that when the distributions are relatively flat ($c < 0.6$) all samplers have comparable performance. However, as the distributions become sharper with several well defined modes ($c \geq 0.6$), the SM sampler significantly outperforms all other samplers. As mentioned above, when the distribution has well defined modes the path from one mode to the other using only local swaps is likely to go through low probability modes. This is the likely cause of the poor performance of the GB and CF samplers as both samplers propose new assignments through local moves. As in the learning to rank experiments, we found the rejection rate for the RP sampler to increase significantly for $N \geq 25$. We were unable to obtain any samples in the allocated time (over 5 mins) from the RP sampler for $N = 50$. Overall, the results further show that the SM method is able to generate higher quality samples faster than the other methods.

Figure 2: Example image pair with $N = 25$. Green lines show the inferred MAP assignment.

## 6 Conclusion

In this paper we introduced a new sampling approach for bipartite matching problems based on a generalization of the Plackett-Luce model. In this approach the matching probabilities at each stage are conditioned on the partial assignment made to that point. This global dependency allows us to define a rich class of proposal distributions that accurately approximate the target distribution. Empirically we found that our method is able to generate good quality samples faster and is less prone to getting stuck in local modes. Future work involves applying the sampler during inference while learning BMP models. We also plan to investigate the relationship between the proposal distribution produced by sequential matching and the target one.

## Footnotes

[1]The proof is in the supplementary material.

[2]Trace and Hellinger distance plots (for both experiments) are in the supplementary material.

# References

[1] A. Bouchard-Cote and M. I. Jordan. Variational inference over combinatorial spaces. In *NIPS*, 2010.

[2] C. Cadena, D. Galvez-Lopez, F. Ramos, J. D. Tardos, and J. Neira. Robust place recognition with stereo cameras. In *IROS*, 2010.

[3] T. S. Caetano, L. Cheng, Q. V. Le, and A. J. Smola. Learning graph matching. In *ICML*, 2009.

[4] O. Chapelle, Y. Chang, and T.-Y. Liu. The Yahoo! Learning to Rank Challenge. 2010.

[5] F. Dellaert, S. M. Seitz, C. E. Thorpe, and S. Thrun. EM, MCMC, and chain flipping for structure from motion with unknown correspondence. *Machine Learning*, 50, 2003.

[6] J.-P. Doignon, A. Pekec, and M. Regenwetter. The repeated insertion model for rankings: Missing link between two subset choice models. *Psychometrika*, 69, 2004.

[7] C. Dwork, R. Kumar, M. Naor, and D. Sivakumar. Rank aggregation methods for the web. In *WWW*, 2001.

[8] B. Huang and T. Jebara. Loopy belief propagation for bipartite maximum weight b-matching. In *AISTATS*, 2007.

[9] J. Huang, C. Guestrin, and L. Guibas. Fourier theoretic probabilistic inference over permutations. *Machine Learning Research*, 10, 2009.

[10] M. Huber and J. Law. Fast approximation of the permanent for very dense problems. In *SODA*, 2008.

[11] M. Jerrum, A. Sinclair, and E. Vigoda. A polynomial-time approximation algorithm for the permanent of a matrix with non-negative entries. 2004.

[12] Q. V. Le and A. Smola. Direct optimization of ranking measures. In *arxiv: 0704.3359*, 2007.

[13] T. Lu and C. Boutilier. Learning Mallows models with pairwise preferences. In *ICML*, 2011.

[14] R. D. Luce. *Individual choice behavior: A theoretical analysis*. Wiley, 1959.

[15] R. M. Neal. Probabilistic inference using Markov Chain Monte Carlo methods. Technical report, University of Toronto, 1993.

[16] C. H. Papadimitriou and K. Steiglitz. *Combinatorial optimization: Algorithms and complexity*. Prentice-Hall, 1982.

[17] J. Petterson, T. S. Caetano, J. J. McAuley, and J. Yu. Exponential family graph matching and ranking. In *NIPS*, 2009.

[18] R. Plackett. The analysis of permutations. *Applied Statistics*, 24, 1975.

[19] T. Qin, T.-Y. Liu, X.-D. Zhang, D.-S. Wang, and H. Li. Global ranking using continuous conditional random fields. In *NIPS*, 2008.

[20] F. Ramos, D. Fox, and H. Durrant-Whyte. CRF-Matching: Conditional random fields for feature-based scan matching. In *Robotics: Science and Systems*, 2007.

[21] D. A. Ross, D. Tarlow, and R. S. Zemel. Learning articulated structure and motion. *International Journal on Computer Vision*, 88, 2010.

[22] R. Salakhutdinov. Learning deep Boltzmann machines using adaptive MCMC. In *ICML*, 2010.

[23] M. Taylor, J. Guiver, S. Robertson, and T. Minka. Softrank: Optimizing non-smooth rank metrics. In *WSDM*, 2008.

[24] W. R. Taylor. Protein structure comparison using bipartite graph matching and its application to protein structure classification. In *Molecular Cell Proteomics*, 2002.

[25] L. G. Valiant. The complexity of computing the permanent. *Theoretical Computer Science*, 8, 1979.

[26] M. N. Volkovs and R. S. Zemel. Boltzrank: Learning to maximize expected ranking gain. In *ICML*, 2009.

[27] Y. Wang, F. Makedon, and J. Ford. A bipartite graph matching framework for finding correspondences between structural elements in two proteins. In *IEBMS*, 2004.

